# Multi-agent Cooperation in Diverse Population Games

**K. Y. Michael Wong, S. W. Lim and Z. Gao**
Hong Kong University of Science and Technology, Hong Kong, China.
{phkywong, swlim, zhuogao}@ust.hk

## Abstract

We consider multi-agent systems whose agents compete for resources by striving to be in the minority group. The agents adapt to the environment by reinforcement learning of the preferences of the policies they hold. Diversity of preferences of policies is introduced by adding random biases to the initial cumulative payoffs of their policies. We explain and provide evidence that agent cooperation becomes increasingly important when diversity increases. Analyses of these mechanisms yield excellent agreement with simulations over nine decades of data.

## 1   Introduction

In the intelligent control of large systems, the multi-agent approach has the advantages of parallelism, robustness, scalability, and light communication overhead [1]. Since it involves many interacting adaptive agents, the behavior becomes highly complex. While a standard analytical approach is to study their steady state behavior described by the Nash equilibria [2], it is interesting to consider the dynamics of how the steady state is approached. Of particular interest is the case of heterogeneous agents, which have diversified preferences in decision making. In such cases, the cooperation of agents becomes very important.

Specifically, we consider the dynamics of a version of large population games which models the collective behavior of agents simultaneously and adaptively competing for limited resources. The game is a variant of the Minority Game, in which the agents strive to make the minority decision, thereby balancing the load distributed between the majority and minority choices [3]. Previous work showed that the system behavior depends on the input dimension of the agents' policies. When the policy dimension is too low, many agents share identical policies, and the system suffers from the *maladaptive* behavior of the agents, meaning that they prematurely rush to adapt to system changes in bursts [4].

Recently, we have demonstrated that a better system efficiency can be attained by introducing diversity [5]. This is done by randomly assigning biases to the initial preference of policies of the agents, so that agents sharing common policies may not adopt them at the same time, and maladaptation is reduced. As a result, the population difference between the majority and minority groups decreases. For typical control tasks such as the distribution of shared resources, this corresponds to a high system efficiency. In contrast to the maladaptive regime, in which agents blindly respond to environmental signals, agent cooperation becomes increasingly important in the diverse regime. Namely, there are fewer agents ad-

justing their policy perferences at each step of the steady state, but there emerges a more coordinated pattern of policy adjustment among them. Hence, it is interesting to study the mechanisms by which they adapt mutually, and their effects on the system efficiency.

In this paper, we explain the cooperative mechanisms which appear successively when the diversity of the agents' preference of policies increases, as recently proposed in [6]. We will provide experimental evidence of these effects, and sketch their analyses which yield excellent agreement with simulations. While we focus on the population dynamics of the Minority Game, we expect that the observed cooperative mechanisms are relevant to reinforcement learning in multi-agent systems more generally.

## 2   The Minority Game

The Minority Game consists of a population of $N$ agents competing selfishly to maximize their individual utility in an environment of limited resources, $N$ being odd [3]. Each agent makes a decision $+$ or $-$ at each time step, and the minority group wins. For typical control tasks such as the resource allocation, the decisions $+$ and $-$ may represent two alternative resources, so that less agents utilizing a resource implies more abundance. The decisions of each agent are prescribed by *policies*, which are binary functions mapping the *history* of the winning bits of the game in the most recent $m$ steps to decisions $+$ or $-$. Hence, $m$ is the memory size. Before the game starts, each agent randomly picks $s$ policies out of the set of $2^D$ policies with replacement, where $D \equiv 2^m$ is the number of input states.

The long-term goal of an agent is to maximize her cumulative payoff, which is the sum of the undiscounted payoffs received during the game history. For the decision $\xi_i(t)$ of agent $i$ at time $t$ ($\xi_i(t) = \pm 1$), the payoff is $-\xi_i(t)G(A(t))$, where $A(t) \equiv \sum_i \xi_i(t)/N$, and $G(A)$ satisfies the property $\text{sign}G(A) = \text{sign}A$. She tries to achieve her goal by choosing at each step, out of her $s$ policies, the most successful one so far, and outputing her decision accordingly. The success of a policy is measured by its cumulative payoff, updated every step irrespective of whether it is adopted or not. This reinforcement learning provides an agent with adaptivity. Though we only consider random policies instead of organized ones, we expect that the model is sufficient to capture the collective behavior of large population games. In this paper, we consider a step payoff function, $G(A) = \text{sign}A$. The cumulative payoffs then take integer values. Note that an agent gains in payoff when she makes a decision opposite to $A(t)$, and loses otherwise, reflecting the winning of the minority group.

It is natural to consider systems with diverse preferences of policies [5]. This means that the initial cumulative payoffs of policies $\alpha$ ($\alpha = 1, \ldots, s - 1$) of agent $i$ with respect to her $s$th policy have random biases $\omega_{i\alpha}$. Diversity is important in reducing the maladaptive behavior of the agents, since otherwise the same policy of all agents accumulates the same payoffs, and would be adopted at the same time. In this paper, we consider the case $s = 2$, and the biases are the sums of $\pm 1$ randomly drawn $R$ times. In particular, when $R$ is not too small, the bias distribution approaches a Gaussian distribution with mean 0 and variance $R$. The ratio $\rho \equiv R/N$ is referred to as the *diversity*. For odd $R$, no two policies have the same cumulative payoffs throughout the process, and the dynamics is deterministic, resulting in highly precise simulation results useful for refined comparison with theories.

The population averages of the decisions oscillate around 0 at the steady state. Since a large difference between the majority and minority populations implies inefficient resource allocation, the inefficiency of the system is often measured by the variance $\sigma^2/N$ of the population making decision $+$, and is given by

$$\frac{\sigma^2}{N} \equiv \frac{N}{4} \langle [A^{\mu^*(t)}(t) - \langle A^{\mu^*(t)}(t) \rangle_t]^2 \rangle_t, \tag{1}$$

where $\langle \ \ \rangle_t$ denotes time average at the steady state. Its dependence on the diversity is

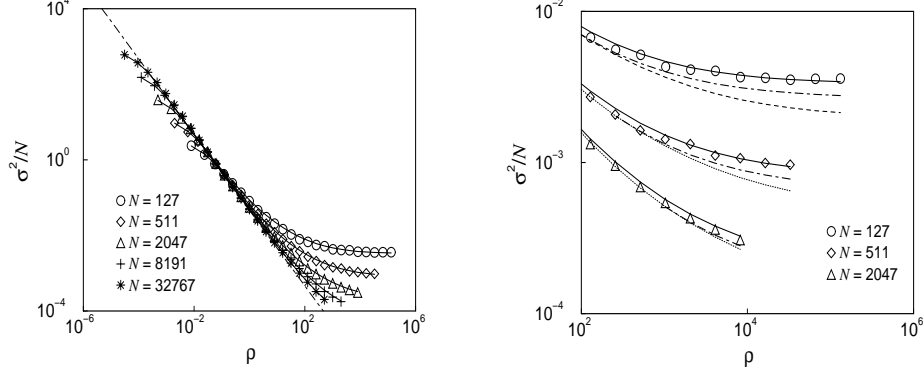

Figure 1: (a) The dependence of the variance of the population making decision $+$ on the diversity at $m = 1$ and $s = 2$. Symbols: simulation results averaged over 1,024 samples of initial conditions. Lines: theory. Dashed-dotted line: scaling prediction. (b) Comparison between simulation results (symbols), theory with kinetic sampling only (dashed lines), one-wait approximation (dash-dotted lines), and many-wait approximation (lines).

shown in Fig. 1. Several modes of agent cooperation can be identified, and explained in the following sections.

## 3 Statistical Cooperation

For each curve with a given $N$ in Fig. 1(a), and besides the first few data points where $\rho \sim N^{-1}$ and $\sigma^2/N \sim N$, the behavior of the variance is dominated by the scaling relation $\sigma^2/N \sim \rho^{-1}$ for $\rho \sim 1$. To interpret this result, we describe the macroscopic dynamics of the system by defining the $D$-dimensional vector $A^\mu(t)$, which is the sum of the decisions of all agents responding to history $\mu$ of their policies, normalized by $N$. While only one of the $D$ components corresponds to the historical state $\mu^*(t)$ of the system, the augmentation to $D$ components is necessary to describe the attractor structure and the transient behavior of the system dynamics.

The key to analysing the system dynamics is the observation that the cumulative payoffs of all policies displace by exactly the same amount when the game proceeds. Hence for a given pair of policies, the profile of the relative cumulative payoff distribution remains unchanged, but the peak position shifts with the game dynamics. Let us consider the change in $A^\mu(t)$ when $\mu$ is the historical state $\mu^*(t)$. We let $S_{\alpha\beta}(\omega)$ be the number of agents holding policies $\alpha$ and $\beta$ (with $\alpha < \beta$), and the bias of $\alpha$ with respect to $\beta$ is $\omega$. If the cumulative payoff of policy $\alpha$ at time $t$ is $\Omega_\alpha(t)$, then the agents holding policies $\alpha$ and $\beta$ make decisions according to policy $\alpha$ if $\omega + \Omega_\alpha(t) - \Omega_\beta(t) > 0$, and policy $\beta$ otherwise. Hence $\omega + \Omega_\alpha(t) - \Omega_\beta(t)$ is referred to as the *preference* of $\alpha$ with respect to $\beta$. At time $t$, the cumulative payoff of policy $\alpha$ changes from $\Omega_\alpha(t)$ to $\Omega_\alpha(t) - \xi_\alpha^\mu \text{sign} A^\mu(t)$, where $\xi_\alpha^\mu$ is the decision of policy $\alpha$ at state $\mu$. Only the *fickle* agents, that is, those agents with preferences on the verge of switching signs, contribute to the change in $A^\mu(t)$, namely, $\omega + \Omega_\alpha(t) - \Omega_\beta(t) = \pm 1$ and $\xi_\alpha^\mu - \xi_\beta^\mu = \pm 2 \text{sign} A^\mu(t)$. Hence we have

$$A^\mu(t+1) - A^\mu(t) = -\text{sign} A^\mu(t) \frac{2}{N} \sum_{\alpha < \beta} \sum_{r=\pm 1} S_{\alpha\beta}(r - \Omega_\alpha(t) + \Omega_\beta(t))$$
$$\times \delta(\xi_\alpha^\mu - \xi_\beta^\mu - 2r \text{sign} A^\mu(t)) \tag{2}$$

where $\delta(n) = 1$ if $n = 0$, and 0 otherwise. In the region where $D \ll \ln N$, we have

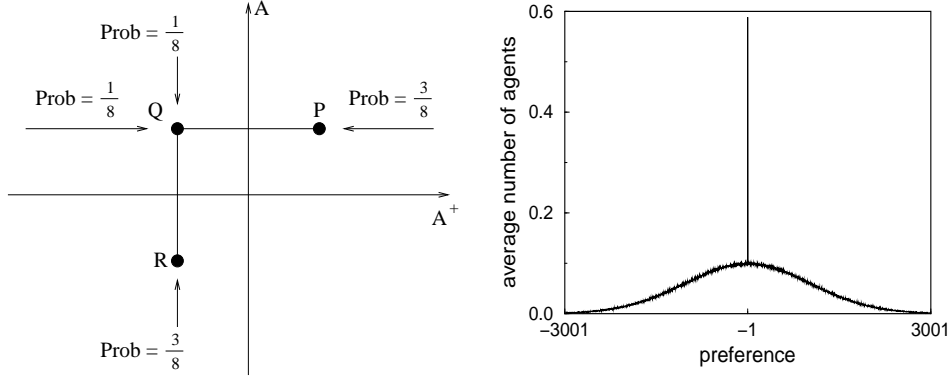

Figure 2: (a) The attractor in the Minority Game with $m = 1$, following the period-4 sequence of P-Q-R-Q in the phase space of $A^+$ and $A^-$. There are 4 approaches to the attractor indicated by the arrows, and the respective probabilities are obtained by considering the detailed dynamics from the different initial positions and states. (b) Experimental evidence of the kinetic sampling effect: steady-state preference dependence of the average number of agents holding the identity policy and its complement, immediately before state Q enters state R, at $\rho = N = 1,023$ and averaged over 100,000 samples of initial conditions.

$S_{\alpha\beta}(\omega) \gg 1$, and Eq. (2) is self-averaging. Following the derivation in [5], we arrive at

$$A^\mu(t+1) = A^\mu(t) - \text{sign} A^\mu(t)\sqrt{\frac{2}{\pi R}}\delta(\mu - \mu^*(t)). \qquad (3)$$

Equation (3) shows that the dynamics proceeds in the direction which reduces the magnitude of the population vector, each time by a step of size $\sqrt{2/\pi R}$. At the steady state, each component oscillates between positive and negative, as shown in the example of $m = 1$ in Fig. 2(a). Due to the maladaptive nature of the dynamics, it never reaches the zero value. As a result, each state is confined in a $D$-dimensional hypercube of size $\sqrt{2/\pi R}$, irrespective of the initial position of the population vector. This confinement enables us to compute the variance of the decisions, given by $\sigma^2/N = f(\rho)/2\pi\rho$, where $f(\rho)$ is a smooth function of $\rho$, which approaches $(1 - 1/4D)/3$ for $\rho \gg 1$. The physical picture of this scaling relation comes from the broadening of the preference distribution due to bias diversity. The fraction of fickle agents at every time step consists of those who have $\pm 1$ preferences, which scales as the height of the bias distribution near its center. Since the distribution is a Gaussian with standard deviation $\sqrt{R}$, the step sizes scale as $1/\sqrt{R}$, and variances $\sigma^2/N$ as $\rho^{-1}$. The scaling relation shows that agent cooperation in this regime is described at the level of statistical distributions of policy preferences, since the number of agents making an adaptive move at each step is sufficiently numerous ($\sim \sqrt{N}$).

## 4   Kinetic Sampling

As shown in Fig. 1(a), $\sigma^2/N$ deviates above the scaling with $\rho^{-1}$ when $\rho \sim N$. To consider the origin of this deviation, we focus in Fig. 2(b) on how the average number of agents, who hold the identity policy with $\xi_\alpha^\mu = \mu$ and its complementary policy $\xi_\beta^\mu = -\mu$, depends on the preference $\omega + \Omega_\alpha - \Omega_\beta$, when the system reaches the steady state in games with $m = 1$. Since the preferences are time dependent, we sample their frequencies at a fixed time, say, immediately before the state changes from Q to R in Fig. 2(a). One would expect that the bias distribution is reproduced. However, we find that a peak exists at $\omega + \Omega_\alpha - \Omega_\beta = -1$.

This value of the preference corresponds to that of the attractor step from Q to R when at state $-$, decision $+$ loses and decision $-$ wins, and $\omega + \Omega_\alpha - \Omega_\beta$ changes from $-1$ to $+1$. The peak at the attractor step shows that its average size is self-organized to be larger than those of the transient steps described by the background distribution.

This effect that favors the cooperation of larger clusters of agents is referred to as the *kinetic sampling* effect. When $\rho \sim N$, $A^\mu(t+1) - A^\mu(t)$ scales as $N^{-1}$ and is no longer self-averaging. Rather, Eq. (2) shows that it is equal to $2/N$ times the number of fickle agents at time $t$, which is Poisson distributed with a mean of $N/\sqrt{2\pi R} = \Delta/2$, where $\Delta \equiv N\sqrt{2/\pi R}$ is the average step size. However, since the attractor is formed by steps which *reverse the sign of $A^\mu$*, the average step size in the attractor is *larger* than that in the transient state, because a long jump in the vicinity of the attractor is more likely to get trapped.

To describe this effect, we consider the probability $P_{\mathrm{att}}(\Delta\mathbf{A})$ of step sizes $\Delta\mathbf{A}$ in the attractor (with $\Delta A^\mu > 0$ for all $\mu$). Assuming that all states of the phase space are equally likely to be accessed, we have $P_{\mathrm{att}}(\Delta\mathbf{A}) = \sum_{\mathbf{A}} P_{\mathrm{att}}(\Delta\mathbf{A}, \mathbf{A})$, where $P_{\mathrm{att}}(\Delta\mathbf{A}, \mathbf{A})$ is the probability of finding the position $\mathbf{A}$ with displacement $\Delta\mathbf{A}$ in the attractor. Consider the example of $m = 1$, where there is only one step along each axis $A^\mu$. The sign reversal condition implies that $P_{\mathrm{att}}(\Delta\mathbf{A}, \mathbf{A}) \propto P_{\mathrm{Poi}}(\Delta\mathbf{A}) \prod_\mu \Theta[-A^\mu(A^\mu + \Delta A^\mu)]$, where $\Theta(x)$ is the step function of $x$, and $P_{\mathrm{Poi}}(\Delta\mathbf{A})$ is the Poisson distribution of step sizes, yielding $P_{\mathrm{att}}(\Delta\mathbf{A}) \propto P_{\mathrm{Poi}}(\Delta\mathbf{A}) \prod_\mu \Delta A^\mu$. We note that the extra factors of $\Delta A^\mu$ favor larger step sizes. Thus, the attractor averages $\langle(\Delta A^\pm)^2\rangle_{\mathrm{att}}$ are given by

$$\langle(\Delta A^\pm)^2\rangle_{\mathrm{att}} = \frac{\langle(\Delta A^\pm)^2\Delta A^+\Delta A^-\rangle_{\mathrm{Poi}}}{\langle\Delta A^+\Delta A^-\rangle_{\mathrm{Poi}}}. \tag{4}$$

There are agents who contribute to both $\Delta A^+$ and $\Delta A^-$, giving rise to their correlations. In Eq. (2), the strategies of the agents contributing to $\Delta A^+$ and $\Delta A^-$ satisfy $\xi_\alpha^+ - \xi_\beta^+ = -2r$ and $\xi_\alpha^- - \xi_\beta^- = 2r$ respectively. Among the agents contributing to $\Delta A^+$, the extra requirement of $\xi_\alpha^- - \xi_\beta^- = 2r$ implies that an average of $1/4$ of them also contribute to $\Delta A^-$. Hence, the number of agents contributing to both steps is a Poisson variable with mean $\Delta/8$, and those exclusive to the individual steps are Poisson variables with mean $3\Delta/8$. This yields, for example,

$$\langle\Delta A^+\Delta A^-\rangle_{\mathrm{Poi}} = \quad \frac{4}{N^2}\sum_{a_0,a_+,a_-} \frac{e^{-\frac{\Delta}{8}}}{a_0!}\left(\frac{\Delta}{8}\right)^{a_0} \frac{e^{-\frac{3\Delta}{8}}}{a_+!}\left(\frac{3\Delta}{8}\right)^{a_+} \frac{e^{-\frac{3\Delta}{8}}}{a_-!}\left(\frac{3\Delta}{8}\right)^{a_-}$$
$$(a_0 + a_+)(a_0 + a_-). \tag{5}$$

Together with similar expressions of the numerator in Eq. (4), we obtain

$$\langle(\Delta A^\pm)^2\rangle_{\mathrm{att}} = \frac{2\Delta^3 + 15\Delta^2 + 20\Delta + 4}{N^2(2\Delta + 1)}. \tag{6}$$

The attractor states are given by $A^\mu = m_\mu/N$ and $m_\mu/N - \Delta A^\mu$, where $m_\mu = 1, 3, \ldots, N\Delta A^\mu - 1$. This yields a variance of

$$\frac{\sigma^2}{N} = \frac{7\langle(N\Delta A^+)^2\rangle_{\mathrm{att}} + 7\langle(N\Delta A^-)^2\rangle_{\mathrm{att}} - 8}{192N}, \tag{7}$$

which gives, on combining with Eq. (6),

$$\frac{\sigma^2}{N} = \frac{14\Delta^3 + 105\Delta^2 + 132\Delta + 24}{96N(2\Delta + 1)}. \tag{8}$$

When the diversity is low, $\Delta \gg 1$, and Eq. (8) reduces to $\sigma^2/N = 7/48\pi\rho$, agreeing with the scaling result of the previous section. When $\rho \sim N$, Eq. (8) has excellent agreement with simulation results, which significantly deviate above the scaling relation.

# 5 Waiting Effect

As shown in Fig. 1(b), $\sigma^2/N$ further deviates above the predictions of kinetic sampling when $\rho \gg N$. To study the origin of this effect, we consider the example of $m = 1$. As shown in Fig. 2(a), the attractor consists of both hops along the $A^\pm$ axes. Analysis shows that only those agents holding the identity policy and its complement can complete both hops after they have adjusted their preferences to $\omega + \Omega_\alpha - \Omega_\beta = \pm 1$. Since there are fewer and fewer fickle agents in the limit $\rho \gg N$, one would expect that a single agent of this type would dominate the game dynamics, and $\sigma^2/N$ would approach $0.25/N$, as also predicted by Eq. (8).

However, attractors having 2 fickle agents are about 10 times more common in the extremely diverse limit. As illustrated in Fig. 3(a) for a typical case, one of the two agents first arrives at the status of $\pm 1$ preference of her policies and stay there waiting. Meanwhile, the preference of the second agent is steadily reduced. Once she has arrived at the status of $\pm 1$ preference of her policies, both agents can then cooperate to complete the dynamics of the attractor. In this example, both agents do not belong to the correct type that can complete the dynamics alone, but waiting is crucial for them to complete the hops in the attractor, even though one would expect that the probability of finding more than one fickle agents at a time step is drastically less than that for one. Thus, the composition of the group of fickle agents is self-organized through this waiting effect, and consequently the step sizes and variance increase above those predicted by kinetic sampling.

The analysis of the waiting effect is lengthy. Here the agents are so diverse that the average step size is approaching 0. At each state in the phase space, the system remains stationary for many time steps, waiting for some agent to reduce the magnitude of her preference until policy switching can take place. For illustration, we sketch the approximation of including up to one wait. As shown in Fig. 2(a), the attractor may be approached from the arm (P or R) or from the corner (Q). Consider the case of the state approaching from P, waiting up to $k$ times at Q to move to R, and ending the transient dynamics thereafter. Then the cumulative payoffs of a policy $\alpha$ can be written as $\Omega_\alpha + \xi_\alpha^+$ at P, $\Omega_\alpha, \ldots, \Omega_\alpha - k\xi_\alpha^-$ at Q and, in the attractor of period 4, repeating the sequence of $\Omega_\alpha - k\xi_\alpha^- - \xi_\alpha^-$ at R, $\Omega_\alpha - k\xi_\alpha^-$ at Q, $\Omega_\alpha - k\xi_\alpha^- + \xi_\alpha^+$ at P, and $\Omega_\alpha - k\xi_\alpha^-$ at Q. The movement of the cumulative payoffs can be conveniently represented by writing $\Omega_\alpha = \sum_\mu k^\mu \xi_\alpha^\mu$, where $k^\mu$ denotes the number of wins minus losses of decision 1 at state $\mu$ in the game history. For $m = 1$, these steps are plotted in the space of $k^+$ and $k^-$ in Fig. 3(b).

The size of each step is $2/N$ times the number of fickle agents at that step, which is Poisson distributed with average $\Delta/2$. The average numbers of agents appearing simultaneously in different steps positioned along the directions $k^+ \pm k^- = \text{constant}$ and $k^\pm = \text{constant}$ are, respectively, $\Delta/8$ and $\Delta/4$, and 0 for other directions. Thus, the average number of agents common in the pairs of steps $\{PQ, QQ_1\}$, $\{QQ_k, QP\}$, $\{QP, QR\}$, $\{PQ, QP\}$ are $\Delta/8$, $\Delta/8$, $\Delta/8$ and $\Delta/4$ respectively. The rest of the combinations of steps are uncorrelated. The number of agents involved in the steps are described in Table 1.

The variance of the step sizes is given by

$$\langle \frac{1}{2}[(\Delta A^+)^2 + (\Delta A^-)^2] \rangle_{\text{att}} = \sum_j P_j \left( \frac{\sum_{i=0,1} \langle \frac{1}{2}[(\Delta A^+)^2 + (\Delta A^-)^2] \Delta A^+ \Delta A^- \rangle_{i,j}}{\sum_{i=0,1} \langle \Delta A^+ \Delta A^- \rangle_{i,j}} \right),$$

(9)

where $j$ = arm or corner. The variance of decisions can then be obtained from Eq. (7). For illustration, we consider the derivation of the Poisson average $\langle \Delta A^+ \Delta A^- \rangle$ for one-wait

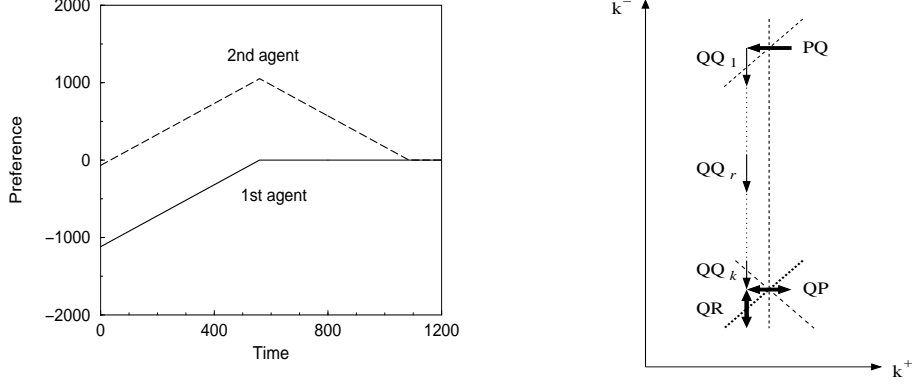

Figure 3: (a) Experimental evidence of the waiting effect: a typical example of the evolution of the preference of the 2 agents switching policies at the attractor in a game with $m = 1$, $N = 127$, and $R = 2^{24} - 1$. The system converges to the attractor at $t = 1,086$. (b) The space of $k^+$ and $k^-$ describing the movement of the cumulative payoffs in the game with $m = 1$. Thick arrows: non-vanishing steps. Thin arrows: waiting steps. Thick double arrows: attractor steps. The dashed lines link those steps that share common agents.

arm approach. Noting that the only non-vanishing steps are PQ, QR and QP, we obtain

$$\langle \Delta A^+ \Delta A^- \rangle_{1,\text{arm}} = \frac{4}{N^2} \sum_{k=1}^{\infty} \langle [1 - \delta(a_i)\delta(a_{\text{turn},1})\delta(a_{\text{cum}})]\delta(a_{\text{turn},1})$$
$$\prod_{r=1}^{k} \delta(a_{\text{wait},r})\delta(a_{\text{turn},2})(a_- + a_0)(a_{\text{cum}} + a_{\text{turn},2} + a_0)\rangle_{\text{Poi}}$$
$$= \frac{4}{N^2} \frac{1}{1 - e^{-\frac{\Delta}{2}}} \left\{ e^{-\frac{\Delta}{2}} \left[ 12 \left(\frac{\Delta}{8}\right)^2 + \frac{\Delta}{8} \right] - e^{-\frac{7\Delta}{8}} \left[ 4 \left(\frac{\Delta}{8}\right)^2 + \frac{\Delta}{8} \right] \right\}. \tag{10}$$

We note that the number $a_0$ accounts for the agents who contribute to both steps in the attractor, and thus can complete the attractor dynamics alone in the extremely diverse limit. On the other hand, the number $a_{\text{cum}}$ arises from the first step PQ arriving at Q. Once present, it will appear in the attractor step QP, irrespective of the duration $k$ of the wait at Q. These $a_{\text{cum}}$ agents can wait to complete the attractor dynamics together with the $a_-$ agents who contribute independently to the step from Q to R, as well as the $a_0$ agents who contribute to both attractor steps. As a result, the average step size increases due to this waiting effect. In the former case, cooperation between individual types of agents becomes indispensable in reaching the steady state behavior.

Other Poisson averages in Eq. (9) can be derived similarly. As shown in Fig. 1(b), the waiting effect causes the variance to increase beyond the kinetic sampling prediction, agreeing with the trend of the simulation results. In particular, the variance approaches $0.34/N$ in the extremely diverse limit, significantly greater than the limit of $0.25/N$ in the absence of waiting effects. Further approximation including multiple waiting steps results in the theoretical curves with excellent agreement with the simulation results, as shown in Fig. 1(b). In the extremely diverse limit, the theoretical predictions approach $0.42/N$, very close to the simulation result of $0.43/N$.

## 6  Conclusion

We have studied the dynamical mechanisms of cooperation, which emerges automatically in a multi-agent system with adaptive agents competing selfishly for finite resources. At low diversity, agent cooperation proceeds at the statistical level, resulting in the scaling relation of the variance with diversity. At high diversity, when kinetic sampling becomes

Table 1: The number of fickle agents in the steps of one wait.

| Label | Steps | No. of agents | Poisson averages |
|---|---|---|---|
| PQ | $\Omega_\alpha + \xi_\alpha^+ \to \Omega_\alpha$ | $a_i + a_{\text{turn},1} + a_{\text{cum}}$ | $\langle a_i \rangle = \Delta/8$, $\langle a_{\text{turn},1} \rangle = \Delta/8$, $\langle a_{\text{cum}} \rangle = \Delta/4$. |
| $QQ_1$ | $\Omega_\alpha \to \Omega_\alpha - \xi_\alpha^-$ | $a_{\text{wait},1} + a_{\text{turn},1}$ | $\langle a_{\text{wait},1} \rangle = 3\Delta/8$. |
| $QQ_r$ | $\Omega_\alpha - (r-1)\xi_\alpha^-$ $\to \Omega_\alpha - r\xi_\alpha^-$ | $a_{\text{wait},r}$ | $\langle a_{\text{wait},r} \rangle = \Delta/2$, $(2 \leq r \leq k-1)$. |
| $QQ_k$ | $\Omega_\alpha - (k-1)\xi_\alpha^-$ $\to \Omega_\alpha - k\xi_\alpha^-$ | $a_{\text{wait},k} + a_{\text{turn},2}$ | $\langle a_{\text{wait},k} \rangle = 3\Delta/8$, $\langle a_{\text{turn},2} \rangle = \Delta/8$. |
| QR | $\Omega_\alpha - k\xi_\alpha^- \to$ $\Omega_\alpha - (k+1)\xi_\alpha^-$ | $a_- + a_0$ | $\langle a_- \rangle = 3\Delta/8$, $\langle a_0 \rangle = \Delta/8$. |
| QP | $\Omega_\alpha - k\xi_\alpha^- \to$ $\Omega_\alpha - k\xi_\alpha^- + \xi_\alpha^+$ | $a_{\text{cum}} + a_{\text{turn},2} + a_0$ | |

significant, we find that the attractor dynamics favors the cooperation of larger clusters of agents. In extremely diverse systems, we further discover a waiting mechanism, when agents who are unable to complete the attractor dynamics alone wait for other agents to collaborate with them. When waiting is present, cooperation between individual types of agents becomes indispensable in reaching the steady state behavior. Together, these mechanisms yield theoretical predictions of the population variance in excellent agreement with simulations over nine decades of data.

We expect that the observed mechanisms of agent cooperation can be found in reinforcement learning of multi-agent systems in general, due to their generic nature. The mechanisms of statistical cooperation, kinetic sampling and waiting illustrate the importance of dynamical considerations in describing the system behavior, and the capability of multi-agent systems to self-organize in their collective dynamics. In particular, it is interesting to note that given enough waiting time, agents with limited abilities can cooperate to achieve dynamics unachievable by individuals. This is relevant to evolutionary approaches to multi-agent control, since it allows limited changes to accumulate into bigger improvements.

### Acknowledgments

We thank C. H. Yeung, Y. S. Ting and B. H. Wang for fruitful discussions. This work is supported by the Research Grant Council of Hong Kong (HKUST6153/01P, HKUST6062/02P) and DAG04/05.SC25.

## References

[1] G. Weiß and S. Sen, Adaption and Learning in Multi-agent Systems, Lecture Notes in Computer Science 1042 (Springer, Berlin, 1995).

[2] E. Rasmusen, Games and Information (Basil Blackwell, Oxford, 2001).

[3] D. Challet and Y. C. Zhang, Emergence of Cooperation and Organization in an Evolutionary Game, Physica A **246**, pp. 407-418 (1997).

[4] R. Savit, R. Manuca, and R. Riolo, Adaptive Competition, Market Efficiency, and Phase Transitions, Phys. Rev. Lett. **82**, pp. 2203-2206 (1999).

[5] K. Y. M. Wong, S. W. Lim, and P. Luo, Diversity and Adaptation in Large Population Games, Int. J. Mod. Phys. B **18**, 2422-2431 (2004).

[6] K. Y. M. Wong, S. W. Lim, and Z. Gao, Dynamical Mehanisms of Adaptation in Multi-agent Systems, Phys. Rev. E **70**, 025103(R) (2004).